# Optimal Brain Surgeon:
# Extensions and performance comparisons

Babak Hassibi*        David G. Stork        Gregory Wolff

**Takahiro Watanabe**
Ricoh California Research Center
2882 Sand Hill Road Suite 115
Menlo Park, CA 94025-7022
and
*Department of Electrical Engineering
105B Durand Hall
Stanford University
Stanford, CA 94305-4055

## Abstract

We extend Optimal Brain Surgeon (*OBS*) — a second-order method for pruning networks — to allow for general error measures, and explore a reduced computational and storage implementation via a dominant eigenspace decomposition. Simulations on nonlinear, noisy pattern classification problems reveal that *OBS* does lead to improved generalization, and performs favorably in comparison with Optimal Brain Damage (*OBD*). We find that the required retraining steps in *OBD* may lead to inferior generalization, a result that can be interpreted as due to injecting noise back into the system. A common technique is to stop training of a large network at the minimum validation error. We found that the test error could be reduced even further by means of *OBS* (but not *OBD*) pruning. Our results justify the $\mathbf{t} \to \mathbf{o}$ approximation used in *OBS* and indicate why retraining in a highly pruned network may lead to inferior performance.

# 1   INTRODUCTION

The fundamental theory of generalization favors simplicity. For a given level of performance on observed data, models with fewer parameters can be expected to perform better on test data. In practice, we find that neural networks with fewer weights typically generalize better than large networks with the same training error. To this end, LeCun, Denker and Solla's (1990) Optimal Brain Damage method (*OBD*) sought to delete weights by keeping the training error as small as possible. Hassibi and Stork (1993) extended *OBD* to include the off-diagonal terms in the network's Hessian, which were shown to be significant and important for pruning in classical and benchmark problems.

*OBD* and Optimal Brain Surgeon (*OBS*) share the same basic approach of training a network to (local) minimum in error at weight $\mathbf{w}^*$, and then pruning a weight that leads to the smallest increase in the training error. The predicted functional increase in the error for a change in full weight vector $\delta\mathbf{w}$ is:

$$\delta E = \underbrace{\left(\frac{\partial E}{\partial \mathbf{w}}\right)^T \cdot \delta\mathbf{w}}_{\approx 0} + \frac{1}{2}\,\delta\mathbf{w}^T \cdot \underbrace{\frac{\partial^2 E}{\partial \mathbf{w}^2}}_{\equiv \mathbf{H}} \cdot \delta\mathbf{w} + \underbrace{O(\|\delta\mathbf{w}\|^3)}_{\approx 0}\,, \tag{1}$$

where $\mathbf{H}$ is the Hessian matrix. The first term vanishes because we are at a local minimum in error; we ignore third- and higher-order terms (Gorodkin et al., 1993). Hassibi and Stork (1993) first showed that the *general* solution for minimizing this function given the constraint of deleting one weight was:

$$\delta\mathbf{w} = -\frac{w_q}{[\mathbf{H}^{-1}]_{qq}}\,\mathbf{H}^{-1} \cdot \mathbf{e}_q \ \text{ and } \ L_q = \frac{1}{2}\,\frac{w_q^2}{[\mathbf{H}^{-1}]_{qq}}\,. \tag{2}$$

Here, $\mathbf{e}_q$ is the unit vector along the $q$th direction in weight space and $L_q$ is the *saliency* of weight $q$ — an estimate of the increase in training error if weight $q$ is pruned and the other weights updated by the left equation in Eq. 2.

# 2   GENERAL ERROR MEASURES AND FISHER'S METHOD OF SCORING

In this section we show that the recursive procedure for computing the inverse Hessian for sum squared errors presented in Hassibi and Stork (1993) generalizes to any twice differentiable distance norm and that the key approximation based on Fisher's method of scoring is still valid.

Consider an arbitrary twice differentiable distance norm $d(\mathbf{t}, \mathbf{o})$ where $\mathbf{t}$ is the desired output (teaching vector) and $\mathbf{o} = F(\mathbf{w}, \mathbf{in})$ the actual output. Given a weight vector $\mathbf{w}$, $F$ maps the input vector $\mathbf{in}$ to the output; the total error over P patterns is $E = \frac{1}{P}\sum_{k=1}^{P} d(\mathbf{t}^{[k]}, \mathbf{o}^{[k]})$. It is straightforward to show that for a single output unit network the Hessian is:

$$\mathbf{H} = \frac{1}{P} \sum_{k=1}^{P} \frac{\partial F(\mathbf{w}, \mathbf{in}^{[k]})}{\partial \mathbf{w}} \cdot \frac{\partial^2 d(\mathbf{t}^{[k]}, \mathbf{o}^{[k]})}{\partial \mathbf{o}^2} \cdot \frac{\partial F^T(\mathbf{w}, \mathbf{in}^{[k]})}{\partial \mathbf{w}} +$$

$$\frac{1}{P} \sum_{k=1}^{P} \frac{\partial d(\mathbf{t}^{[k]}, \mathbf{o}^{[k]})}{\partial \mathbf{o}} \cdot \frac{\partial^2 F(\mathbf{w}, \mathbf{in}^{[k]})}{\partial \mathbf{w}^2} . \qquad (3)$$

The second term is of order $O(\|\mathbf{t} - \mathbf{o}\|)$; using Fisher's method of scoring (Sever & Wild, 1989), we set this term to zero. Thus our Hessian reduces to:

$$\mathbf{H} = \frac{1}{P} \sum_{k=1}^{P} \frac{\partial F(\mathbf{w}, \mathbf{in}^{[k]})}{\partial \mathbf{w}} \cdot \frac{\partial^2 d(\mathbf{t}^{[k]}, \mathbf{o}^{[k]})}{\partial \mathbf{o}^2} \cdot \frac{\partial F^T(\mathbf{w}, \mathbf{in}^{[k]})}{\partial \mathbf{w}} . \qquad (4)$$

We define $\mathbf{X}_k \equiv \frac{\partial F(\mathbf{w}, \mathbf{in}^{[k]})}{\partial \mathbf{w}}$ and $a_k \equiv \frac{\partial^2 d(\mathbf{t}^{[k]}, \mathbf{o}^{[k]})}{\partial \mathbf{o}^2}$, and following the logic of Hassibi and Stork (1993) we can easily show that the recursion for computing the inverse Hessian becomes:

$$\mathbf{H}_{k+1}^{-1} = \mathbf{H}_k^{-1} - \frac{\mathbf{H}_k^{-1} \cdot \mathbf{X}_{k+1} \cdot \mathbf{X}_{k+1}^T \cdot \mathbf{H}_k^{-1}}{\frac{P}{a_k} + \mathbf{X}_{k+1}^T \cdot \mathbf{H}_k^{-1} \cdot \mathbf{X}_{k+1}} , \quad \mathbf{H}_0^{-1} = \alpha^{-1}\mathbf{I}, \text{ and} \quad \mathbf{H}_P^{-1} = \mathbf{H}^{-1} ,$$

$$(5)$$

where $\alpha$ is a small parameter — effectively a weight decay constant. Note how different error measures $d(\mathbf{t}, \mathbf{o})$ scale the gradient vectors $\mathbf{X}_k$ forming the Hessian (Eq. 4). For the squared error $d(\mathbf{t}, \mathbf{o}) = (\mathbf{t} - \mathbf{o})^2$, we have $a_k = 1$, and all gradient vectors are weighted equally. The cross entropy or Kullback-Leibler distance,

$$d(\mathbf{t}, \mathbf{o}) = \mathbf{o} \log \frac{\mathbf{o}}{\mathbf{t}} + (1 - \mathbf{o}) \log \frac{(1 - \mathbf{o})}{(1 - \mathbf{t})} , \qquad 0 \leq \mathbf{o}, \mathbf{t} \leq 1 \qquad (6)$$

yields $a_k = \frac{1}{\mathbf{o}^{[k]}(1 - \mathbf{o}^{[k]})}$. Hence if $\mathbf{o}^{[k]}$ is close to zero or one, $\mathbf{X}_k$ is given a large weight in the Hessian; conversely, the smallest value of $a_k$ occurs when $\mathbf{o}^{[k]} = 1/2$. This is desirable and makes great intuitive sense, since in the cross entropy norm the value of $\mathbf{o}^{[k]}$ is interpreted as the probability that the $k$th input pattern belongs to a particular class, and therefore we give large weight to $\mathbf{X}_k$ whose class we are most certain and small weight to those which we are least certain.

## 3   EIGENSPACE DECOMPOSITION

Although *OBS* has been shown to be a powerful method for small and intermediate sized networks — Hassibi, Stork and Wolff (1993) applied *OBS* successfully to NETtalk — its use in larger problems is difficult because of large storage and computation requirements. For a network of $n$ weights, simply storing the Hessian requires $O(n^2/2)$ elements and $O(Pn^2)$ computations are needed for each pruning step. Reducing this computational burden requires some type of approximation. Since *OBS* uses the inverse of the Hessian, any approximation to *OBS* will at some level reduce to an approximation of $\mathbf{H}$. For instance *OBD* uses a diagonal approximation; magnitude-based methods use an isotropic approximation; and dividing the network into subsets (e.g., hidden-to-output and input-to-hidden) corresponds to the less-restrictive *block* diagonal approximation. In what follows we explore the dominant eigenspace decomposition of the inverse Hessian as our approximation. It should be remembered that all these are subsets of the full *OBS* approach.

### 3.1 Theory

The dominant eigendecomposition is the best low-rank approximation of a matrix (in an induced 2-norm sense). Since the largest eigenvalues of $\mathbf{H}^{-1}$ are the smallest eigenvalues of $\mathbf{H}$, this method will, roughly speaking, be pruning weights in the approximate nullspace of $\mathbf{H}$. Dealing with a low rank approximation of $\mathbf{H}^{-1}$ will drastically reduce the storage and computational requirements.

Consider the eigendecomposition of $\mathbf{H}$:

$$\mathbf{H} = (\mathbf{U}_S \mathbf{U}_N) \begin{pmatrix} \mathbf{\Sigma}_S & 0 \\ 0 & \mathbf{\Sigma}_N \end{pmatrix} \begin{pmatrix} \mathbf{U}_S^* \\ \mathbf{U}_N^* \end{pmatrix} = \mathbf{U}_S \mathbf{\Sigma}_S \mathbf{U}_S^* + \mathbf{U}_N \mathbf{\Sigma}_N \mathbf{U}_N^*, \qquad (7)$$

where $\mathbf{\Sigma}_S$ contains the largest eigenvalues of $\mathbf{H}$ and $\mathbf{\Sigma}_N$ the smallest ones. (We use the subscripts $S$ and $N$ to loosely connote signal and noise.) The dimension of the noise subspace is typically $m \ll n$. $\mathbf{U}_S$ and $\mathbf{U}_N$ are $n \times (n-m)$ and $n \times m$ matrices that span the dominant eigenspace of $\mathbf{H}$ and $\mathbf{H}^{-1}$, and * denotes matrix transpose and complex conjugation. If, as suggested above, we restrict the weight prunings to lie in $\mathbf{U}_N$, we obtain the following saliency and full weight change when removing the $q$th weight:

$$\bar{L}_q = \frac{1}{2} \frac{w_q^2}{\mathbf{e}_q^T \cdot \mathbf{U}_N \cdot \mathbf{\Sigma}_N^{-1} \cdot \mathbf{U}_N^* \cdot \mathbf{e}_q} \qquad (8)$$

$$\delta \bar{\mathbf{w}} = -\frac{w_q}{\mathbf{e}_q^T \cdot \mathbf{U}_N \cdot \mathbf{\Sigma}_N^{-1} \cdot \mathbf{U}_N^* \cdot \mathbf{e}_q} \mathbf{\Sigma}_N^{-1} \mathbf{U}_N^* \mathbf{e}_q , \qquad (9)$$

where we have used 'bars' to indicate that these are approximations to Eq. 2. Note now that we need only to store $\mathbf{\Sigma}_N$ and $\mathbf{U}_N$, which have roughly $nm$ elements. Likewise the computation required to estimate $\mathbf{\Sigma}_N$ and $\mathbf{U}_N$ is $O(Pnm)$.

The bound on $\bar{L}_q$ is:

$$L_q \le \bar{L}_q \le L_q + 2 \frac{L_q \bar{L}_q}{w_q^2} \cdot \frac{1}{\underline{\sigma}(s)}, \qquad (10)$$

where $\underline{\sigma}(s)$ is the smallest eigenvalue of $\mathbf{\Sigma}_S$. Moreover if $\underline{\sigma}(s)$ is large enough so that $\underline{\sigma}(s) > \frac{1}{[\mathbf{H}^{-1}]_{qq}}$ we have the following simpler form:

$$L_q \le \bar{L}_q \le \frac{L_q}{1 - \frac{1}{[\mathbf{H}^{-1}]_{qq}\underline{\sigma}(s)}}. \qquad (11)$$

In either case Eqs. 10 and 11 indicate that the larger $\underline{\sigma}(s)$ is, the tighter the bounds are. Thus if the subspace dimension $m$ is such that the eigenvalues in $\mathbf{U}_S$ are large, then we will have a good approximation.

LeCun, Simard and Pearlmutter (1993) have suggested a method that can be used to estimate the smallest eigenvectors of the Hessian. However, for *OBS* (as we shall see) it is best to use the Hessian with the $\mathbf{t} \to \mathbf{o}$ approximation, and their method is not appropriate.

## 3.2 Simulations

We pruned networks trained on the three Monk's problems (Thrun et al., 1991) using the full *OBS* and a 5-dimensional eigenspace version of *OBS*, using the validation error rate for stopping criterion. (We chose a 5-dimensional subspace, because this reduced the computational complexity by an order of magnitude.) The Table shows the number of weights obtained. It is clear that this eigenspace decomposition was not particularly successful. It appears as though the the off-diagonal terms in **H** *beyond* those in the eigenspace are important, and their omission leads to bad pruning. However, this warrants further study.

|  | unpruned | OBS | 5-d eigenspace |
|---|---|---|---|
| Monk1 | 58 | 14 | 28 |
| Monk2 | 39 | 16 | 27 |
| Monk3 | 39 | 4 | 11 |

# 4 OBS/OBD COMPARISON

General criteria for comparing pruning methods do not exist. Since such methods amount to assuming a particular prior distribution over the parameters, the empirical results usually tell us more about the problem space, than about the methods themselves. However, for two methods, such as *OBS* and *OBD*, which utilize the same cost function, and differ only in their approximations, empirical comparisons can be informative. Hence, we have applied both *OBS* and *OBD* to several problems, including an artificially generated statistical classification task, and a real-world copier voltage control problem. As we show below, the *OBS* algorithm usually results in better generalization performance.

## 4.1 MULTIPLE GAUSSIAN PRIORS

We created a two-catagory classification problem with a five-dimensional input space. Category A consisted of two Gaussian distributions with mean vectors $\mu_{A1} = (1, 1, 0, 1, .5)$ and $\mu_{A2} = (0, 0, 1, 0, .5)$ and covariances $\Sigma_{A1} = Diag[0.99, 1.0, 0.88, 0.70, 0.95]$ and $\Sigma_{A2} = Diag[1.28, 0.60, 0.52, 0.93, 0.93]$ while category B had means $\mu_{B1} = (0, 1, 0, 0, .5)$ and $\mu_{B2} = (1, 0, 1, 1, .5)$ and covariances $\Sigma_{B1} = Diag[0.84, 0.68, 1.28, 1.02, 0.89]$ and $\Sigma_{B2} = Diag[0.52, 1.25, 1.09, 0.64, 1.13]$. The networks were feedforward with 5 input units, 9 hidden units, and a single output unit (64 weights total). The training and the test sets consisted of 1000 patterns each, randomly chosen from the equi-probable categories. The problem was a difficult one: even with the somewhat large number of weights it was not possible to obtain less than 0.15 squared error per training pattern. We trained the networks to a local error minimum and then applied *OBD* (with retraining after each pruning step using backpropagation) as well as *OBS*.

Figure 1 (left) shows the training errors for the network as a function of the number of remaining weights during pruning by *OBS* and by *OBD*. As more weights are pruned the training errors for both *OBS* and *OBD* typically increase. Comparing the two graphs for the first pruned weights, the training error for *OBD* and *OBS* are roughly equal, after which the training error of *OBS* is less until the 24th weight

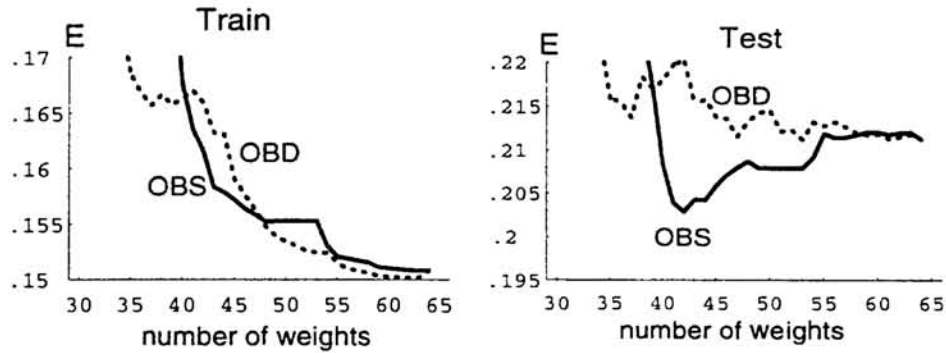

Figure 1: *OBS* and *OBD* training error on a sum of Gaussians prior pattern classification task as a function of the number of weights in the network. (Pruning proceeds right to left.) *OBS* pruning employed $\alpha = 10^{-6}$ (cf., Eq. 5); *OBD* employed 60 retraining epochs after each pruning.

is removed. The reason *OBD* training is initially slightly better is that the network was not at an exact local minimum; indeed in the first few stages the training error for *OBD* actually becomes *less* than its original value. (Training exhaustively to the true local minimum took prohibitively long.) In contrast, due to the $\mathbf{t} \to \mathbf{o}$ approximation *OBS* tries to keep the network response close to where it was, even if that isn't the minimum $\mathbf{w}^*$. We think it plausible that if the network were at an exact local minimum *OBS* would have had virtually identical performance.

Since *OBD* is using retraining the only reason why *OBS* can outperform after the first steps is that *OBD* has removed an incorrect weight, due to its diagonal approximation. (The reason *OBS* behaves poorly after removing 24 weights — a radically pruned net — may be that the second-order approximation breaks down at this point.) We can see that the minimum on test error occurs before this breakdown, meaning that the failed approximation (Fig. 2) does not affect our choice of the optimal network, at least for this problem.

The most important and interesting result is the *test* error for these pruned networks (Figure 1, right). The test error for *OBD* does not show any consistent behaviour, other than the fact that on the average it generally goes *up*. This is contrary to what one would expect of a pruning algorithm. It seems that the retraining phase works against the pruning process, by tending to reinforce overfitting, and to reinject the training set noise. For *OBS*, however, the test error consistently decreases until after removing 22 weights a minimum is reached, because the $\mathbf{t} \to \mathbf{o}$ approximation avoids reinjecting the training set noise.

## 4.2   OBS/OBD PRUNING AND "STOPPED" NETWORKS

A popular method of avoiding overfitting is to stop training a large net when the validation error reaches a minimum. In order to explore whether pruning could improve the performance on such a "stopped" network (i.e., not at $\mathbf{w}^*$), we monitored the test error for the above problem and recorded the weights for which a minimum on the test set occured. We then applied *OBS* and *OBD* to this network.

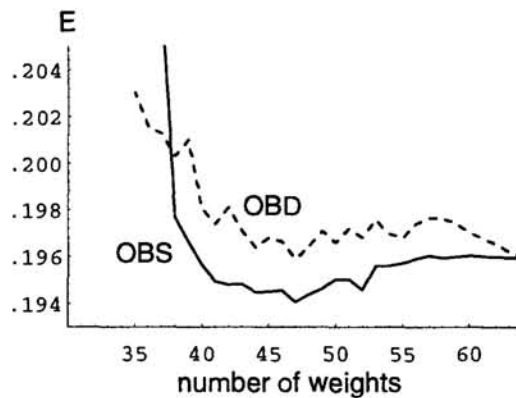

Figure 2: A 64-weight network was trained to minimum validation error on the Gaussian problem — *not* **w**\* — and then pruned by *OBD* and by *OBS*. The test error on the resulting network is shown. (Pruning proceeds from right to left.) Note especially that even though the network is far from **w**\*, *OBS* leads lower test error over a wide range of prunings, even through *OBD* employs retraining.

The results shown in Figure 2 indicate that with *OBS* we were able to reduce the test error, and this reached a minimum after removing 17 weights. *OBD* was not able to consistently reduce the test error.

This last result and those from Fig. 2 have important consequences. There are no universal stopping criteria based on theory (for the reasons mentioned above), but it is a typical practice to use validation error as such a criterion. As can be seen in Figure 2, the test error (which we here consider a validation error) consistantly decreases to a unique miniumum for pruning by *OBS*. For the network pruned (and continuously retrained) by *OBD*, there is no such structure in the validation curves. There seems to be no reliable clue that would permit the user to know when to stop pruning.

## 4.3    COPIER CONTROL APPLICATION

The quality of an image produced by a copier is dependent upon a wide variety of factors: time since last copy, time since last toner cartridge installed, temperature, humidity, overall graylevel of the source document, etc. These factors interact in a highly non-linear fashion, so that mathematical modelling of their interrelationships is difficult. Morita et al. (1992) used backpropagation to train an 8–4–8 network (65 weights) on real-world data, and managed to achieve an RMS voltage error of 0.0124 on a critical control plate. We pruned his network with both *OBD* with retraining as well as with *OBS*. When the network was pruned by *OBD* with retraining, the test error continually increased (erratically) such that at 34 remaining weights, the RMS error was 0.023. When also we pruned the original net by *OBS*, and the test error gradually decreased such that at the same number of weights the test error was 0.012 — significantly lower than that of the net pruned by *OBD*.

## 5   CONCLUSIONS

We compared pruning by *OBS* and by *OBD* with retraining on a difficult non-linear statistical pattern recognition problem and found that *OBS* led to lower generalization error. We also considered the widely used technique of training large nets to minimum validation error. To our surprise, we found that subsequent pruning by *OBS* lowered generalization error, thereby demonstrating that such networks still have overfitting problems. We have found that the dominant eigenspace approach to *OBS* leads to poor performance. Our simulations support the claim that the $t \rightarrow o$ approximation used in *OBS* avoids reinjecting training set noise into the network. In contrast, including such $t - o$ terms in *OBS* reinjects training set noise and degrades generalization performance, as does retraining in *OBD*.

### Acknowledgements

Thanks to T. Kailath for support of B.H. through grants AFOSR 91-0060 and DAAL03-91-C-0010. Address reprint requests to Dr. Stork: stork@crc.ricoh.com.

### References

J. Gorodkin, L. K. Hansen, A. Krogh, C. Svarer and O. Winther. (1993) A quantitative study of pruning by Optimal Brain Damage. *International Journal of Neural Systems* 4(2) 159-169.

B. Hassibi & D. G. Stork. (1993) Second order derivatives for network pruning: Optimal Brain Surgeon. In S. J. Hanson, J. D. Cowan and C. L. Giles (eds.), *Advances in Neural Information Processing Systems 5*, 164-171. San Mateo, CA: Morgan Kaufmann.

B. Hassibi, D. G. Stork & G. Wolff. (1993) Optimal Brain Surgeon and general network pruning. *Proceedings of ICNN 93, San Francisco* 1 IEEE Press. 293-299.

Y. LeCun, J. Denker & S. Solla. (1990) Optimal Brain Damage. In D. Touretzky (ed.), *Advances in Neural Information Processing Systems 2*, 598-605. San Mateo, CA: Morgan Kaufmann.

Y. LeCun, P. Simard & B. Pearlmutter. (1993) Automatic learning rate maximization by on-line estimation of the Hessian's eigenvectors. In S. J. Hanson, J. D. Cowan & C. L. Giles (eds.), *Advances in Neural Information Processing Systems 5*, 156-163. San Mateo, CA: Morgan Kaufmann.

T. Morita, M. Kanaya, T. Inagaki, H. Murayama & S. Kato. (1992) Photo-copier image density control using neural network and fuzzy theory. *Second International Workshop on Industrial Fuzzy Control & Intelligent Systems* December 2-4, College Station, TX, 10.

S. Thrun and 23 co-authors. (1991) The Monk's Problems — A performance comparison of different learning algorithms. CMU-CS-91-197 Carnegie-Mellon University Dept. of Computer Science Technical Report.